# Efficient Bayesian Parameter Estimation in Large Discrete Domains

**Nir Friedman**
Hebrew University
nir@cs.huji.ac.il

**Yoram Singer**
AT&T Labs
singer@research.att.com

## Abstract

We examine the problem of estimating the parameters of a multinomial distribution over a large number of discrete outcomes, most of which do not appear in the training data. We analyze this problem from a Bayesian perspective and develop a hierarchical prior that incorporates the assumption that the observed outcomes constitute only a small subset of the possible outcomes. We show how to *efficiently* perform *exact* inference with this form of hierarchical prior and compare it to standard approaches.

## 1 Introduction

One of the most important problems in statistical inference is *multinomial* estimation: Given a past history of observations independent trials with a discrete set of outcomes, predict the probability of the next trial. Such estimators are the basic building blocks in more complex statistical models, such as prediction trees [1, 12, 11], hidden Markov models [9] and Bayesian networks [3, 6]. The roots of multinomial estimation go back to Laplace's work in the 18th century [7].

In Bayesian theory, the classic approach to multinomial estimation is via the use of the *Dirichlet* distribution (see for instance [4]). Laplace's "law of succession" and other common methods can be derived using Bayesian inference with the Dirichlet distribution as a prior distribution. The Dirichlet distribution has several properties that are useful in statistical inference. In particular, estimates with Dirichlet priors are *consistent* (the estimate converges with probability one to the true distribution), *conjugate* (the posterior distribution is also a Dirichlet distribution), and can be computed efficiently (queries of interest have a closed-form solution). Furthermore, theoretical studies of online prediction of individual sequences show that prediction using Dirichlet priors is competitive with *any* other prior distribution (see for instance [2] and the references therein).

Unfortunately, in some key applications, Dirichlet priors are unwieldy. These applications are characterized by several features: (a) The set of possible outcomes is extremely large, and often not known in advance. (b) The number of training examples is small compared

to the number of possible outcomes. (c) The outcomes that have positive probability constitute a relatively small subset of the possible outcomes; this subset, however, is not known in advance. In these situations, predictions based on a Dirichlet priors tend to assign most of the probability mass to outcomes that were not seen in the training set.

For example, consider a natural language application, where outcomes are words drawn from an English dictionary, and the problem is predicting the probability of words that follow a particular word, say "Bosnia". If we do not have any prior knowledge, we can consider any word in the dictionary as a possible candidate. Yet, our knowledge of language would lead us to believe that in fact, only few words, such as "Herzegovina", should naturally follow the word "Bosnia". Furthermore, even in a large corpus, we do not expect to see many training examples that involve this phrase. As another example consider the problem of estimating the parameters of a discrete dynamical system. Here the task is to find a distribution over the states that can be reached from a particular state $s$ (possibly after the system receives an external control signal). Again, in many domains it is natural to assume that the system is sparse: only a small subset of states is reachable from any state.

In this paper, we present a Bayesian treatment of this problem using an *hierarchical* prior that averages over an exponential number of hypotheses each of which represents a subset of the feasible outcomes. Such a prior was previously used in a specific context of online prediction using suffix tree transducers [11]. As we show, although this prior involves exponentially many hypotheses, we can *efficiently* perform predictions. Moreover, our approach allows us to deal with countably infinite number of outcomes.

## 2   Dirichlet priors

Let $X$ be a random variable that can take $L$ possible values from a set $\Sigma$. Without loss of generality, let $\Sigma = \{1, \ldots L\}$. We are given a training set $D$ that contains the outcomes of $N$ independent draws $x^1, \ldots, x^N$ of $X$ from an unknown multinomial distribution $P^*$. We denote by $N_i$ be the number of occurrences of the symbol $i$ in the training data. The *multinomial estimation* problem is to find a good approximation for $P^*$.

This problem can be stated as the problem of predicting the outcome $x^{N+1}$ given $x^1, \ldots, x^N$. Given a prior distribution over the possible multinomial distributions, the Bayesian estimate is:

$$P(x^{N+1} \mid x^1, \ldots, x^N, \xi) = \int P(x^{N+1} \mid \theta, \xi) P(\theta \mid x^1, \ldots, x^N, \xi) d\theta \qquad (1)$$

where $\theta = \langle \theta_1, \ldots, \theta_L \rangle$ is a vector that describes possible values of the (unknown) probabilities $P^*(1), \ldots, P^*(L)$, and $\xi$ is the "context" variable that denotes all other assumptions about the domain. (We consider particular contexts in the next section.)

The posterior probability of $\theta$ can rewritten using Bayes law as:

$$P(\theta \mid x^1, \ldots, x^N, \xi) \propto P(x^1, \ldots, x^N \mid \theta, \xi) P(\theta \mid \xi) = P(\theta \mid \xi) \prod_i \theta_i^{N_i} \qquad (2)$$

The family of *Dirichlet* distributions is *conjugate* to the multinomial distribution. That is, if the prior distribution is from this family, so is the posterior. A Dirichlet prior for $X$ is specified by *hyperparameters* $\alpha_1, \ldots, \alpha_L$, and has the form:

$$P(\theta \mid \xi) = \frac{\Gamma(\sum_i \alpha_i)}{\prod_i \Gamma(\alpha_i)} \prod_i \theta_i^{\alpha_i - 1} \qquad \left( \sum_i \theta_i = 1 \text{ and } \theta_i \geq 0 \text{ for all } i \right)$$

where $\Gamma(x) = \int_0^\infty t^{x-1} e^{-t} dt$ is the *gamma* function. Given a Dirichlet prior, the initial prediction for each value of $X$ is $P(X^1 = i \mid \xi) = \int \theta_i P(\theta \mid \xi) d\theta = \alpha_i / \sum_j \alpha_j$. It is

easy to see that, if the prior is a Dirichlet prior with hyperparameters $\alpha_1, \ldots, \alpha_L$, then the posterior is a Dirichlet with hyperparameters $\alpha_1 + N_1, \ldots, \alpha_L + N_L$. Thus, we get that the prediction for $X^{N+1}$ is $P(X^{N+1} = i \mid x^1, \ldots, x^N, \xi) = (\alpha_i + N_i)/\sum_j(\alpha_j + N_j)$. We can think of the hyperparameters $\alpha_i$ as the number of "imaginary" examples in which we saw outcome $i$. Thus, the ratio between hyperparameters corresponds to our initial assessment of the relative probability of the corresponding outcomes. The total weight of the hyperparameters represent our confidence (or entrenchment) in the prior knowledge. As we can see, if this weight is large, our estimates for the parameters tend to be further off from the empirical frequencies observed in the training data.

## 3  Hierarchical priors

We now describe structured priors that capture our uncertainty about the set of "feasible" values of $X$. We define a random variable $V$ that takes values from the set $2^\Sigma$ of possible subsets of $\Sigma$. The intended semantics for this variable is that $\theta_i > 0$ iff $i \in V$.

Clearly, the hypothesis $V = \Sigma'$ (for $\Sigma' \subseteq \Sigma$) is consistent with training data only if $\Sigma'$ contains all the indices $i$ for which $N_i > 0$. We denote by $\Sigma^o$ the set of observed symbols. That is, $\Sigma^o = \{i : N_i > 0\}$, and we let $k^o = |\Sigma^o|$.

Suppose we know the value of $V$. Given this assumption, we can define a Dirichlet prior over possible multinomial distributions $\theta$ if we use the same hyper-parameter $\alpha$ for each symbol in $V$. Formally, we define the prior:

$$P(\theta|V) = \frac{\Gamma(|V|\alpha)}{\Gamma(\alpha)^{|V|}} \prod_{i \in V} \theta_i^{\alpha-1} \quad \left(\sum_i \theta_i = 1, \forall i, \theta_i \geq 0, \text{ and } \theta_i = 0 \text{ for all } i \notin V\right) \quad (3)$$

Using Eq. (2), we have that:

$$P(X^{N+1} = i \mid x^1, \ldots, x^n, V) = \begin{cases} \frac{\alpha + N_i}{|V|\alpha + N} & \text{if } i \in V \\ 0 & \text{otherwise} \end{cases} \quad (4)$$

Now consider the case where we are uncertain about the actual set of feasible outcomes. We construct a two tiered prior over the values of $V$. We start with a prior over the size of $V$, and then assume that all sets of the same cardinality have the same prior probability. We let the random variable $S$ denote the cardinality of $V$. We assume that we are given a distribution $P(S = k)$ for $k = 1, \ldots, L$. We define the prior over sets to be:

$$P(V \mid S = k) = \binom{L}{k}^{-1} \quad (5)$$

We now examine how to compute the posterior predictions given this hierarchical prior. Let $D$ denote the training data $x^1, \ldots, x^N$. Then it is easy to verify that

$$P(X^{N+1} = i \mid D) = \sum_k \frac{\alpha + N_i}{k\alpha + N} \sum_{V, |V| = k, i \in V} P(V \mid D) \quad (6)$$

Let us now examine which sets $V$ actually contribute to this sum.

First, we note that sets that do not contain $\Sigma^o$ have zero posterior probability, since they are inconsistent with the observed data. Thus, we can examine only sets $V$ that contain $\Sigma^o$. Second, as we noted above, $P(D \mid V)$ is the same for all sets of cardinality $k$ that contain $\Sigma^o$. Moreover, by definition the prior for all these sets is the same. Using Bayes rule, we conclude that $P(V \mid D)$ is the same for all sets of size $k$ that contain $\Sigma^o$. Thus, we can

simplify the inner summation in Eq. (6), by multiplying the number of sets in the score of the summation by the posterior probability of such sets.

There are two cases. If $i \in \Sigma^o$, then any set $V$ that has non-zero posterior probability for $i$ appears in the sum. Thus, in this case we can write:

$$P(X^{N+1} = i \mid D) = \sum_k \frac{\alpha + N_i}{k\alpha + N} P(S = k \mid D) \qquad \text{if } i \in \Sigma^o$$

If $i \notin \Sigma^o$, then we need to estimate the fraction of subsets of $V$ with non-zero posterior that contain $i$. This leads to an equation similar to the one above, but with a correction for this fraction. By symmetry all unobserved outcomes have the same posterior probability. Thus, we can simply divide the mass that was not assigned to the observed outcomes among the unseen symbols.

Notice that the single term in Eq. (3) that depends on $N_i$ can be moved outside the summation. Thus, to make predictions, we only need to estimate the quantity:

$$C(D, L) = \sum_{k=k^o}^{L} \frac{k^o \alpha + N}{k\alpha + N} P(k \mid D)$$

and then

$$P(X^{N+1} = i \mid D) = \begin{cases} \frac{\alpha + N_i}{k^o \alpha + N} C(D, L) & \text{if } i \in \Sigma^o \\ \frac{1}{n - k^o}(1 - C(D, L)) & \text{if } i \notin \Sigma^o \end{cases}$$

We can therefore think of $C(D, L)$ as scaling factor that we apply to the Dirichlet prediction that assumes that we have seen all of the feasible symbols. The quantity $1 - C(D, L)$ is the probability mass assigned to *novel* (i.e., unseen) outcomes. Using properties of Dirichlet priors we get the following characterization of $C(D, L)$.

**Proposition 3.1:** $P(S = k \mid D) = \dfrac{m_k}{\sum_{k' \geq k^o} m_k}$ *where* $m_k = P(S = k) \dfrac{k!}{(k - k^o)!} \cdot \dfrac{\Gamma(k\alpha)}{\Gamma(k\alpha + N)}$.

**Proof:** To compute $C(D, L)$, we need to compute $P(S = k \mid D)$. Using Bayes rule, we have that

$$P(k \mid D) = \frac{P(D \mid S = k) P(S = k)}{\sum_{k'} P(D \mid S = k') P(S = k')} \tag{7}$$

By introduction of variables, we have that:

$$P(D \mid S = k) = \sum_{V \supseteq \Sigma^o, |V| = k} P(D \mid V) P(V \mid S = k).$$

Using standard properties of Dirichlet priors, we have that if $\Sigma^o \subseteq V$, then

$$P(D|V) = \frac{\Gamma(|V|\alpha)}{\Gamma(|V|\alpha + N)} \prod_{i \in V^o} \frac{\Gamma(\alpha + N_i)}{\Gamma(\alpha)} \tag{8}$$

Now, using Eq. (8) and (5), we get that if $\Sigma^o \subseteq V$, and $k = |V|$, then

$$P(D \mid V) P(V \mid S = k) = \binom{L}{k}^{-1} \frac{\Gamma(k\alpha)}{\Gamma(k\alpha + N)} \Gamma(\alpha)^{-k^o} \prod_{i \in \Sigma^o} \Gamma(\alpha + N_i). \tag{9}$$

Thus,

$$\begin{aligned} P(D \mid S = k) &= \binom{L - k^o}{k - k^o} \binom{L}{k}^{-1} \frac{\Gamma(k\alpha)}{\Gamma(k\alpha + N)} \Gamma(\alpha)^{-k^o} \prod_{i \in \Sigma^o} \Gamma(\alpha + N_i) \\ &= \left[ \frac{(L - k^o)!}{L!} \Gamma(\alpha)^{-k^o} \prod_{i \in \Sigma^o} \Gamma(\alpha + N_i) \right] \frac{k!}{(k - k^o)!} \cdot \frac{\Gamma(k\alpha)}{\Gamma(k\alpha + N)} \end{aligned} \tag{10}$$

he following

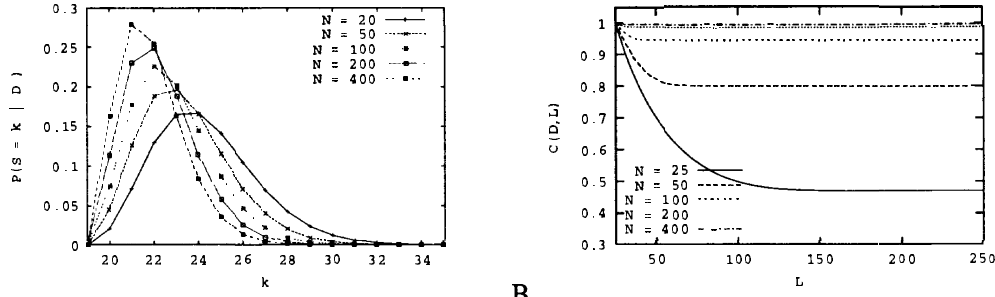

Figure 1: Left: Illustration of the posterior distribution $P(S \mid D)$ for different values of $N$, with $k^o = 20, L = 100, \alpha = .25$, and $P(S = k) \propto 0.25^k$. Right: Illustration showing the change in $C(D, L)$ for different values of $N$, with $k^o = 25, \alpha = 1$, and $P(S = k) \propto 0.9^k$.

The term in the square brackets does not depend on the choice of $k$. Thus, it cancels out when plug Eq. (10) in Eq.(7). The desired equality follows directly. ∎

From the above proposition we immediately get that

$$C(D, L) = \left( \sum_{k=k^o}^{L} \frac{k^o \alpha + N}{k\alpha + N} m_k \right) \left( \sum_{k' \geq k^o} m_k \right)^{-1}. \tag{11}$$

Note that $P(S = k \mid D)$ and $C(D, L)$ depend only on $k^o$ and $N$ and does not depend on the distribution of counts among the $k^o$ observed symbols. Also note that when $N$ is sufficiently larger than $k^o$ (and this depends on the choice of $\alpha$), then the term $\frac{k!}{(k-k^o)!} \cdot \frac{\Gamma(k\alpha)}{\Gamma(k\alpha+N)}$ is much smaller than 1. This implies that the posterior for larger sets decays rapidly. We can see this behavior on the left hand side of Figure 1 that shows the posterior distribution of $P(S \mid D)$ for different dataset sizes.

## 4 Unbounded alphabets

By examining the analytic form of $C(D, L)$, we see that the dependency on $L$ is expressed only in the number of terms in the summation. If the terms $m_k$ vanish for large $k$, then $C(D, L)$ becomes insensitive to the exact size of the alphabet. We can see this behavior on the right hand side of Figure 1, which shows $C(D, L)$ as a function of $L$. As we can see, when $L$ is close to $k^o$, then $C(D, L)$ is close to 1. As $L$ grows, $C(D, L)$ asymptotes to a value that depends on $N$ and $k^o$ (as well as $\alpha$ and the prior $P(S = k)$).

This discussion suggests that we can apply our prior in cases where we do not know $L$ in advance. In fact, we can assume that $L$ is unbounded. That is, $\Sigma$ is isomorphic to $\{1, 2, \ldots\}$. Assume that we assign the prior $P(S = k)$ for each choice of $L$, and that $\lim_{L \to \infty} P(S = k)$ exists for all $k$. We define $C(D, \infty) = \lim_{L \to \infty} C(D, L)$. We then use for prediction the term $P(X^{N+1} = i \mid D) = \frac{\alpha + N_i}{k^o \alpha + N} C(D, \infty)$.

For this method to work, we have to ensure that $C(D, \infty)$ is well defined; that is, that the limit exists. Two such cases are identified by the following proposition (proof omitted).

**Proposition 4.1:** *If $P(S = k)$ is exponentially decreasing in $k$ or if $\alpha \geq 1$ and $P(S = k)$ is polynomially decreasing in $k$, then $C(D, \infty)$ is well-defined.*

In practice we evaluate $C(D, \infty)$ by computing successive values of (the logarithm of) $m_k$, until we reach values that are significantly smaller than the largest value beforehand.

| Method | Perplexity | | |
| --- | --- | --- | --- |
| | Observed | Novel | Overall |
| A $(\frac{N_i}{N+r})$ | 28.19 | 141.7 | 28.20 |
| B (Approximated Good-Turing) | 28.15 | 802.7 | 28.19 |
| Sparse-Multinomial (Poly) | 27.97 | 3812.9 | 28.02 |
| Sparse-Multinomial (Exp) | 27.97 | 3913.1 | 28.03 |

Table 1: Perplexity results on heterogeneous character data.

Since $m_k$ is exponentially decaying, we can ignore the mass in the tail of the sequence. As we can see from the right hand side of Figure 1, there is not much difference between the prediction using a large $L$, and unbounded one.

## 5   Empirical evaluation

We used the proposed estimation method to construct statistical models for predicting the probability of characters in the context of the previously observed character. Such models, often referred to as bigram models, are of great interest in applications such as optical character recognition and text compression. We tested two of prior distributions for the alphabet size $P_0(S = k)$: an exponential prior, $P_0(S = k) \propto \beta^k$, and a polynomial prior, $P_0(S = k) \propto k^{-\beta}$. The training and test material were derived from various archives and included different types of files such C programs, core dumps, and ascii text files. The alphabet for the algorithm consists of all the (ascii and non-ascii) 256 possible characters. The training data consisted of around 170 mega bytes and for testing we used 35 mega bytes.

Each model we compared had to assign a probability to any character. If a character was not observed in the context of the previous character, the new character is assigned the probability of the total mass of novel events. We compared our approach with two estimation techniques that have been shown to perform well on natural data sets [13]. The first estimates the probability of a symbol $i$ in the context of a given word as $\frac{N_i}{N+r}$ where $r$ is the number of different characters observed at given context (the previous character). The second method, based on an approximation of the Good-Turing estimation scheme [5], estimates the probability of a symbol $i$ as $\frac{(1-f_1/N)N_i}{N}$, where $f_1$ is the number of different characters that have been observed only once at for the given context (for more details see [13]). For evaluation we used the perplexity which is simply the exponentiation of the average log-loss on the test data. Table 1 summarizes the average test-set perplexity for observed characters, novel events, and the overall perplexity. In the experiments we fixed $\alpha = 1/2$ for the parameters of the Dirichlet priors and $\beta = 2$ for the exponentially and polynomially decaying priors of the alphabet size.

One can see from the table that predictions using sparse-multinomials achieve the lowest overall perplexity. (The differences are statistically significant due to the size of the data.) The performance based on the two different priors for the alphabet size is comparable. The results indicate the all the leverage in using sparse-multinomials for prediction is due to more accurate predictions for observed events. Indeed, the perplexity of novel events using sparse-multinomials is much higher than when using either method $A$ or $B$. Put another way, our approach prefers to "sacrifice" events with low probability (novel events) and suffer high loss in favor of more accurate predictions for frequently occurring events. The net effect is a lower overall perplexity.

# 6 Discussion

In this paper we presented a Bayesian approach for the problem of estimating the parameters of a multinomial source over a large alphabet. Our method is based on hierarchical priors. We clearly identify the assumptions made by these priors. Given these assumptions, prediction reduces to probabilistic inference. Our main result is showing how to perform this inference *exactly* in an efficient manner. Among the numerous techniques that have been used for multinomial estimation the one proposed by Ristad [10] is the closest to ours. Though the methodology used by Ristad is substantially different than ours, his method can been seen as a special case of sparse-multinomials with $\alpha$ set to 1 and with a specific prior on alphabet sizes. The main advantage of these choices is a simpler inference procedure. This simplicity comes at the price of losing flexibility. In addition, our method explicitly represents the posterior distribution. Hence, it is more suitable for tasks, such as stochastic sampling, where an explicit representation of the approximated distribution is required. Finally, our method can be combined with other Bayesian approaches for language modeling such as the one proposed by MacKay and Peto [8], and with Bayesian approaches for learning complex models such as Bayesian networks [6].

**Acknowledgments** We are grateful to Fernando Pereira and Stuart Russell for discussions related to this work. This work was done while Nir Friedman was at U.C. Berkeley and supported by ARO under grant number DAAH04-96-1-0341 and by ONR under grant number N00014-97-1-0941.

# References

[1] W. Buntine. Learning classification trees. In *Artificial Intelligence Frontiers in Statistics*. Chapman & Hall, 1993.

[2] B.S. Clarke and A.R. Barron. Jeffrey's prior is asymptotically least favorable under entropic risk. *J. Stat. Planning and Inference*, 41:37–60, 1994.

[3] G. F. Cooper and E. Herskovits. A Bayesian method for the induction of probabilistic networks from data. *Machine Learning*, 9:309–347, 1992.

[4] M. H. DeGroot. *Optimal Statistical Decisions*. McGraw-Hill, 1970.

[5] I.J. Good. The population frequencies of species and the estimation of population parameters. *Biometrika*, 40(3):237–264, 1953.

[6] D. Heckerman, D. Geiger, and D. M. Chickering. Learning Bayesian networks: The combination of knowledge and statistical data. *Machine Learning*, 20:197–243, 1995.

[7] P.S. Laplace. *Philosophical Essay on Probabilities*. Springer-Verlag, 1995.

[8] D.J.C. MacKay and L. Peto. A hierarchical Dirichlet language model. *Natural Language Eng.*, 1(3):1–19, 1995.

[9] L. R. Rabiner and B. H. Juang. An introduction to hidden Markov models. *IEEE ASSP Mag.*, 3(1):4–16, 1986.

[10] E. Ristad. A natural law of succession. Tech. Report CS-TR-495-95, Princeton Univ., 1995.

[11] Y. Singer. Adaptive mixtures of probabilistic transducers. *Neur. Comp.*, 9(8):1711–1734, 1997.

[12] F.M.J. Willems, Y.M. Shtarkov, and T.J. Tjalkens. The context tree weighting method: basic properties. *IEEE Trans. on Info. Theory*, 41(3):653–664, 1995.

[13] I.H. Witten and T.C. Bell. The zero-frequency problem: estimating the probabilities of novel events in adaptive text compression. *IEEE Trans. on Info. Theory*, 37(4):1085–1094, 1991.
